# Estimating the Bayes Risk from Sample Data

**Robert R. Snapp\* and Tong Xu**
Computer Science and Electrical Engineering Department
University of Vermont
Burlington, VT 05405

## Abstract

A new nearest-neighbor method is described for estimating the Bayes risk of a multiclass pattern classification problem from sample data (e.g., a classified training set). Although it is assumed that the classification problem can be accurately described by sufficiently smooth class-conditional distributions, neither these distributions, nor the corresponding prior probabilities of the classes are required. Thus this method can be applied to practical problems where the underlying probabilities are not known. This method is illustrated using two different pattern recognition problems.

## 1 INTRODUCTION

An important application of artificial neural networks is to obtain accurate solutions to pattern classification problems. In this setting, each pattern, represented as an $n$-dimensional feature vector, is associated with a discrete pattern class, or state of nature (Duda and Hart, 1973). Using available information, (e.g., a statistically representative set of labeled feature vectors $\{(\mathbf{x}_i, \ell_i)\}$, where $\mathbf{x}_i \in \mathbf{R}^n$ denotes a feature vector and $\ell_i \in \mathbf{L} = \{\omega_1, \omega_2, \ldots, \omega_C\}$, its correct pattern class), one desires a function (e.g., a neural network classifier) that assigns new feature vectors to pattern classes with the smallest possible misclassification cost.

If the classification problem is stationary, such that the patterns from each class are generated according to known probability distributions, then it is possible to construct an optimal classifier that assigns each pattern to a class with minimal expected risk. Although our method can be generalized to problems in which different types of classification errors incur different costs, we shall simplify our discussion by assuming that all errors are equal. In this case, a *Bayes classifier* assigns each feature vector to a class with maximum posterior probability. The expected risk of this classifier, or *Bayes risk* then reduces to the probability of error

$$R_B = \int_S \left[ 1 - \sup_{\ell \in \mathbf{L}} P(\ell|\mathbf{x}) \right] f(\mathbf{x})\, d\mathbf{x}, \tag{1}$$

(Duda and Hart, 1973). Here, $P(\ell|\mathbf{x})$ denotes the posterior probability of class $\ell$ conditioned on observing the feature vector $\mathbf{x}$, $f(\mathbf{x})$ denotes the unconditional mixture density of the feature vector $\mathbf{x}$, and $\mathcal{S} \subset \mathbf{R}^n$ denotes the probability-one support of $f$.

Knowing how to estimate the value of the Bayes risk of a given classification problem with a specific input representation, may facilitate the design of more accurate classifiers. For example, since the value of $R_B$ depends upon the set of features chosen to represent each pattern (e.g., the significance of the input units in a neural network classifier), one might compare estimates of the Bayes risk for a number of different feature sets, and then select the representation that yields the smallest value. Unfortunately, it is necessary to know the explicit probability distributions to evaluate (1). Thus with the possible exception of trivial examples, the Bayes risk cannot be determined exactly for practical classification problems.

Lacking the means to evaluate the Bayes risk exactly, motivates the development of statistical estimators of $R_B$. In this paper, we use a recent asymptotic analysis of the finite-sample risk of the $k$-nearest-neighbor classifier to obtain a new procedure for estimating the Bayes risk from sample data. Section 2 describes the $k$-nearest-neighbor algorithm, and briefly describes how estimates of its finite-sample risk have been used to estimate $R_B$. Section 3 describes how a recent asymptotic analysis of the finite-sample risk can be applied to obtain new statistical estimators of the Bayes risk. In Section 4 the $k$-nearest-neighbor algorithm is used to estimate the Bayes risk of two example problems. Section 5 contains some concluding remarks.

## 2   THE $k$-NEAREST-NEIGHBOR CLASSIFIER

Due to its analytic tractability, and its nearly optimal performance in the large sample limit, the $k$-nearest-neighbor classifier has served as a useful framework for estimating the Bayes risk from classified samples. Recall, that the $k$-nearest-neighbor algorithm (Fix and Hodges, 1951) classifies an $n$-dimensional feature vector $\mathbf{x}$ by consulting a reference sample of $m$ correctly classified feature vectors $\mathcal{X}_m = \{(\mathbf{x}_i, \ell_i) : i = 1, \ldots m\}$. First, the algorithm identifies the $k$ nearest neighbors of $\mathbf{x}$, i.e., the $k$ feature vectors within $\mathcal{X}_m$ that lie closest to $\mathbf{x}$ with respect to a given metric. Then, the classifier assigns $\mathbf{x}$ to the most frequent class label represented by the $k$ nearest neighbors. (A variety of procedures can be used to resolve ties.) In the following, $C$ denotes the number of pattern classes.

The *finite-sample risk* of this algorithm, $R_m$, equals the probability that the $k$-nearest-neighbor classifier assigns $\mathbf{x}$ to an incorrect class, averaged over all input vectors $\mathbf{x}$, and all $m$-samples, $\mathcal{X}_m$. The following properties have been shown to be true under weak assumptions:

**Property 1** *(Cover and Hart, 1967): For fixed $k$,*
$$R_m \to R_\infty(k), \quad as \quad m \to \infty$$
*with*
$$R_B \leq R_\infty(1) \leq R_B\left(2 - \frac{C}{C-1}R_B\right). \tag{2}$$

**Property 2** *(Devroye, 1981): If $k \geq 5$, and $C = 2$, then there exist universal constants $\alpha = 0.3399\cdots$, and $\beta = 0.9749\cdots$ such that $R_\infty(k)$ is bounded by*
$$R_B \leq R_\infty(k) \leq (1 + a_k)R_B, \quad where \quad a_k = \frac{\alpha\sqrt{k}}{k - 3.25}\left(1 + \frac{\beta}{\sqrt{k-3}}\right).$$
*More generally, if $C = 2$, then*
$$R_B \leq R_\infty(k) \leq \left(1 + \sqrt{\frac{2}{k}}\right)R_B. \tag{3}$$

By the latter property, this algorithm is said to be *Bayes consistent* in that for any $\epsilon > 0$, it is possible to construct a $k$-nearest-neighbor classifier such that $|R_m - R_B| < \epsilon$ if $m$ and $k$ are sufficiently large. Bayes consistency is also evident in other nonparametric pattern classifiers.

Several methods for estimating $R_B$ from sample data have previously been proposed, e.g., (Devijver, 1985), (Fukunaga, 1985), (Fukunaga and Hummels, 1987), (Garnett and Yau, 1977), and (Loizou and Maybank, 1987). Typically, these methods involve constructing sequences of $k$-nearest neighbor classifiers, with increasing values of $k$ and $m$. The misclassification rates are estimated using an independent test sample, from which upper and lower bounds to $R_B$ are obtained. Because these experiments are necessarily performed with finite reference samples, these bounds are often imprecise. This is especially true for problems in which $R_m$ converges to $R_\infty(k)$ at a slow rate. In order to remedy this deficiency, it is necessary to understand the manner in which the limit in Property 1 is achieved. In the next section we describe how this information can be used to construct new estimators for the Bayes risk of sufficiently smooth classification problems.

## 3   NEW ESTIMATORS OF THE BAYES RISK

For a subset of multiclass classification problems that can be described by probability densities with uniformly bounded partial derivatives up through order $N + 1$ (with $N \geq 2$), the finite-sample risk of a $k$-nearest-neighbor classifier that uses a weighted $L_p$ metric can be represented by the truncated asymptotic expansion

$$R_m = R_\infty(k) + \sum_{j=2}^{N} c_j m^{-j/n} + O\left(m^{-(N+1)/n}\right), \tag{4}$$

(Psaltis, Snapp, and Venkatesh, 1994), and (Snapp and Venkatesh, 1995). In the above, $n$ equals the dimensionality of the feature vectors, and $R_\infty(k), c_2, \ldots, c_N$, are the expansion coefficients that depend upon the probability distributions that define the pattern classification problem.

This asymptotic expansion provides a parametric description of how the finite-sample risk $R_m$ converges to its infinite sample limit $R_\infty(k)$. Using a large sample of classified data, one can obtain statistical estimates of the finite-sample risk $\hat{R}_m$ for different values of $m$. Specifically, let $\{m_i\}$ denote a sequence of $M$ different sample sizes, and select fixed values for $k$ and $N$. For each value of $m_i$, construct an ensemble of $k$-nearest-neighbor classifiers, i.e., for each classifier construct a random reference sample $\mathcal{X}_{m_i}$ by selecting $m_i$ patterns with replacement from the original large sample. Estimate the empirical risk of each classifier in the ensemble with an independently drawn set of "test" vectors. Let $\hat{R}_{m_i}$ denote the average empirical risk of the $i$-th ensemble. Then, using the resulting set of data points $\{(m_i, \hat{R}_{m_i})\}$, find the values of the coefficients $R_\infty(k)$, and $c_2$ through $c_N$, that minimizes the sum of the squares:

$$\sum_{i=1}^{M}\left(\hat{R}_{m_i} - R_\infty(k) - \sum_{j=2}^{N} c_j m_i^{-j/n}\right)^2 \tag{5}$$

Several inequalities can then be used obtain approximations of $R_B$ from the estimated value of $R_\infty(k)$. For example, if $k = 1$, then Cover and Hart's inequality in Property 1 implies that

$$\frac{R_\infty(1)}{2} \leq R_B \leq R_\infty(1).$$

To enable an estimate of $R_B$ with precision $\epsilon$, choose $k > 2/\epsilon^2$, and estimate $R_\infty(k)$ by the above method. Then Devroye's inequality (3) implies

$$R_\infty(k) - \epsilon \leq R_\infty(k)(1 - \epsilon) \leq R_B \leq R_\infty(k).$$

## 4  EXPERIMENTAL RESULTS

The above procedure for estimating $R_B$ was applied to two pattern recognition problems. First consider the synthetic, two-class problem with prior probabilities $P_1 = P_2 = 1/2$, and normally distributed, class-conditional densities

$$f_\ell(\mathbf{x}) = \frac{1}{(2\pi)^{n/2}} e^{-\frac{1}{2}\left((x_1+(-1)^\ell)^2 + \sum_{i=2}^n x_i^2\right)},$$

for $\ell = 1$ and 2. Pseudorandom labeled feature vectors $(\mathbf{x}, \ell)$ were numerically generated in accordance with the above for dimensions $n = 1$ and $n = 5$. Twelve sample sizes between 10 and 3000 were examined. For each dimension and sample size the risks $R_m$ of many independent $k$-nearest-neighbor classifiers with $k = 1, 7$, and 63 were empirically estimated. (Because the asymptotic expansion (4) does not accurately describe the very small sample behavior of the $k$-nearest-neighbor classifier, sample sizes smaller than $2k$ were not included in the fit.)

Estimates of the coefficients in (5) for six different fits appear in the first equation of each cell in the third and fourth columns of Table 1. For reference, the second column contains the values of $R_\infty(k)$ that were obtained by numerically evaluating an exact integral expression (Cover and Hart, 1967). Estimates of the Bayes risk appear in the second equation of each cell in the third and fourth columns. Cover and Hart's inequality (2) was used for the experiments that assumed $k = 1$, and Devroye's inequality (3) was used if $k \geq 7$. For this problem, formula (1) evaluates to $R_B = (1/2)\,\mathrm{erfc}(1/\sqrt{2}) = 0.15865$.

Table 1: Estimates of the model coefficients and Bayes error for a classification problem with two normal classes.

| $k$ | $R_\infty(k)$ | $n = 1$ $(N = 2)$ | $n = 5$ $(N = 6)$ |
|---|---|---|---|
| 1 | 0.2248 | $R_m = 0.2287 + \dfrac{0.6536}{m^2}$ <br> $R_B = 0.172 \pm 0.057$ | $R_m = 0.2287 + \dfrac{0.1121}{m^{2/5}} + \dfrac{0.2001}{m^{4/5}} - \dfrac{0.0222}{m^{6/5}}$ <br> $R_B = 0.172 \pm 0.057$ |
| 7 | 0.1746 | $R_m = 0.1744 + \dfrac{4.842}{m^2}$ <br> $R_B = 0.152 \pm 0.023$ | $R_m = 0.1700 + \dfrac{0.2218}{m^{2/5}} - \dfrac{1.005}{m^{4/5}} + \dfrac{3.782}{m^{6/5}}$ <br> $R_B = 0.148 \pm 0.022$ |
| 63 | 0.1606 | $R_m = 0.1606 + \dfrac{20.23}{m^2}$ <br> $R_B = 0.157 \pm 0.004$ | $R_m = 0.1595 + \dfrac{0.1002}{m^{2/5}} - \dfrac{1.426}{m^{4/5}} + \dfrac{10.96}{m^{6/5}}$ <br> $R_B = 0.156 \pm 0.004$ |

The second pattern recognition problem uses natural data; thus the underlying probability distributions are not known. A pool of $2^{22}$ classified multispectral pixels were was extracted from a seven band satellite image. Each pixel was represented by five spectral components, $\mathbf{x} = (x_1, \ldots, x_5)$, each in the range $0 \leq x_\nu \leq 255$. (Thus, $n = 5$.) The class label of each pixel was determined by one of the remaining spectral components, $0 \leq y \leq 255$. Two pattern classes were then defined: $\omega_1 = \{y < \theta\}$, and $\omega_2 = \{y \geq \theta\}$, where $\theta$ was a predetermined threshold. (This particular problem was chosen to test the feasibility of this method. In future work, we will examine more interesting pixel classification problems.)

Table 2: Coefficients that minimize the squared error fit for different $N$. Note that $c_3 = 0$ and $c_5 = 0$ in (2) if $n \geq 4$ (Psaltis, Snapp, and Venkatesh, 1994).

| $N$ | $R_\infty(1)$ | $c_2$ | $c_4$ | $c_6$ |
|---|---|---|---|---|
| 2 | 0.0757133 | 0.126214 | | |
| 4 | 0.0757846 | 0.124007 | 0.0132804 | |
| 6 | 0.0766477 | 0.0785847 | 0.689242 | $-2.68818$ |

With $k = 1$, a large number of Bernoulli trials (e.g., 200—1000) were performed for each value of $m_i$. Each trial began by constructing a reference sample of $m_i$ classified pixels chosen at random from the pool. The risk of each reference sample was then estimated by classifying $t$ pixels with the nearest-neighbor algorithm under a Euclidean metric. Here, the $t$ pixels, with $2000 \leq t \leq 20000$, were chosen independently, with replacement, from the pool. The risk $\hat{R}_{m_i}$ was then estimated as the average risk of each reference sample of size $m_i$. (The number of experiments performed for each value of $m_i$, and the values of $t$, were chosen to ensure that the variance of $\hat{R}_{m_i}$ was sufficiently small, less than $10^{-4}$ in this case.) This process was repeated for $M = 33$ different values of $m_i$ in the range $100 \leq m_i \leq 15000$. Results of these experiments are displayed in Table 2 and Figure 1 for three different values of $N$. Note that the robustness of the fit begins to dissolve, for this data, at $N = 6$, either the result of overfitting, or insufficient smoothness in the underlying probability distributions. However, the estimate for $R_\infty(1)$ appears to be stable. For this classification problem, we thus obtain $R_B = 0.0568 \pm 0.0190$.

## 5   CONCLUSION

The described method for estimating the Bayes risk is based on a recent asymptotic analysis of the finite-sample risk of the $k$-nearest-neighbor classifier (Snapp and Venkatesh, 1995). Representing the finite-sample risk as a truncated asymptotic series enables an efficient estimation of the infinite-sample risk $R_\infty(k)$ from the classifier's finite-sample behavior. The Bayes risk can then be estimated by the Bayes consistency of the $k$-nearest-neighbor algorithm. Because such finite-sample analyses are difficult, and consequently rare, this new method has the potential to evolve into a useful algorithm for estimating the Bayes risk. Further improvements in efficiency may be obtained by incorporating principles of optimal experimental design, cf., (Elfving, 1952) and (Federov, 1972).

It is important to emphasize, however, that the validity of (4) rests on several rather strong smoothness assumptions, including a high-degree of differentiability of the class-conditional probability densities. For problems that do not satisfy these conditions, other finite-sample descriptions need to be constructed before this method can be applied. Nevertheless, there is much evidence that nature favors smoothness. Thus, these restrictive assumptions may still be applicable to many important problems.

**Acknowledgments**

The work reported here was supported in part by the National Science Foundation under Grant No. NSF OSR-9350540 and by Rome Laboratory, Air Force Material Command, USAF, under grant number F30602-94-1-0010.

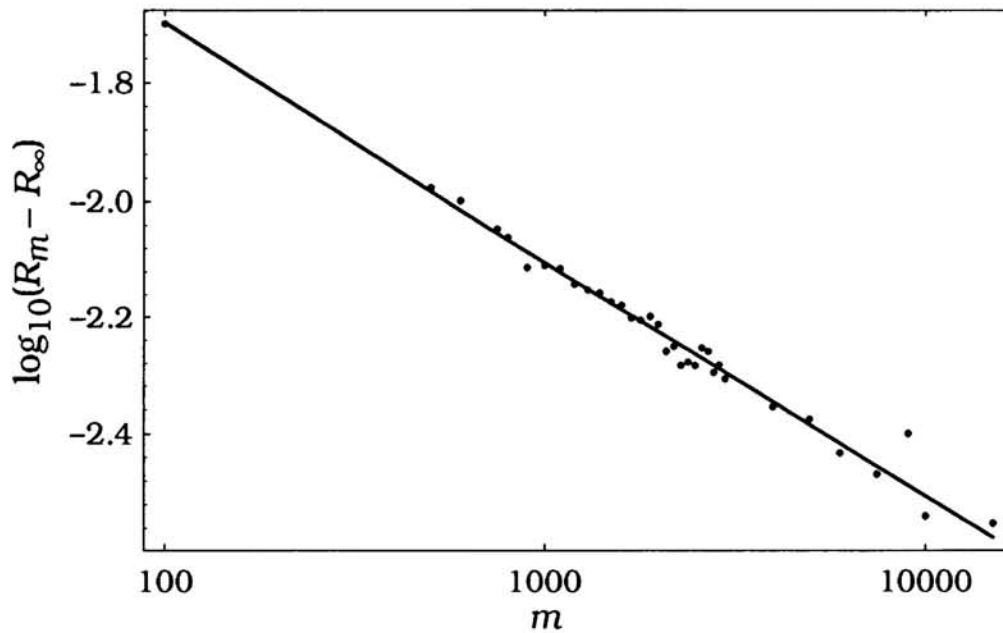

Figure 1: The best fourth-order ($N = 4$) fit of Eqn. (5) to 33 empirical estimates of $\hat{R}_m$, for a pixel classification problem obtained from a multispectral Landsat image. Using $R_\infty = 0.0758$, the fourth-order fit, $R_m = 0.0758 + 0.124m^{-2/5} + 0.0133m^{-4/5}$, is plotted on a log-log scale to reveal the significance of the $j = 2$ term.

## Footnotes

\* E-mail:snapp@emba.uvm.edu

## References

T. M. Cover and P. E. Hart, "Nearest neighbor pattern classification," *IEEE Trans. Inform. Theory*, vol. IT–13, 1967, pp. 21–27.

P. A. Devijver, "A multiclass, $k - NN$ approach to Bayes risk estimation," *Pattern Recognition Letters*, vol. 3, 1985, pp. 1–6.

L. Devroye, "On the asymptotic probability of error in nonparametric discrimination," *Annals of Statistics*, vol. 9, 1981, pp. 1320–1327.

R. O. Duda and P. E. Hart, *Pattern Classification and Scene Analysis*. New York, New York: John Wiley & Sons, 1973.

G. Elfving, "Optimum allocation in linear regression theory," *Ann. Math. Statist.*, vol. 23, 1952, pp. 255–262.

V. V. Federov, *Theory of Optimal Experiments*, New York, New York: Academic Press, 1972.

E. Fix and J. L. Hodges, "Discriminatory Analysis: Nonparametric Discrimination: Consistency Properties," from *Project 21–49–004, Report Number 4*, UASF School of Aviation Medicine, Randolf Field, Texas, 1951, pp. 261–279.

K. Fukunaga, "The estimation of the Bayes error by the $k$-nearest neighbor approach," in L. N. Kanal and A. Rosenfeld (ed.), *Progress in Pattern Recognition*, vol. 2, Elesvier Science Publishers B.V. (North Holland), 1985, pp. 169–187.

K. Fukunaga and D. Hummels, "Bayes error estimation using Parzen and $k$-NN procedures," *IEEE Transactions on Pattern Analysis and Machine Intelligence*, vol. 9, 1987, pp. 634–643.

J. M. Garnett, III and S. S. Yau, "Nonparametric estimation of the Bayes error of feature extractors using ordered nearest neighbor sets," *IEEE Transactions on Computers*, vol. 26, 1977, pp. 46–54.

G. Loizou and S. J. Maybank, "The nearest neighbor and the Bayes error rate," *IEEE Transactions on Pattern Analysis and Machine Intelligence*, vol. 9, 1987, pp. 254–262.

D. Psaltis, R. R. Snapp, and S. S. Venkatesh, "On the finite sample performance of the nearest neighbor classifier," *IEEE Trans. Inform. Theory*, vol. IT–40, 1994, pp. 820—837.

R. R. Snapp and S. S. Venkatesh, "$k$ Nearest Neighbors in Search of a Metric," 1995, (submitted).
